# Throttling Poisson Processes

**Uwe Dick**    **Peter Haider**    **Thomas Vanck**    **Michael Brückner**    **Tobias Scheffer**
University of Potsdam
Department of Computer Science
August-Bebel-Strasse 89, 14482 Potsdam, Germany
{uwedick,haider,vanck,mibrueck,scheffer}@cs.uni-potsdam.de

## Abstract

We study a setting in which Poisson processes generate sequences of decision-making events. The optimization goal is allowed to depend on the *rate* of decision outcomes; the rate may depend on a potentially long backlog of events and decisions. We model the problem as a Poisson process with a throttling policy that enforces a data-dependent rate limit and reduce the learning problem to a convex optimization problem that can be solved efficiently. This problem setting matches applications in which damage caused by an attacker grows as a function of the rate of unsuppressed hostile events. We report on experiments on abuse detection for an email service.

## 1  Introduction

This paper studies a family of decision-making problems in which discrete events occur on a continuous time scale. The time intervals between events are governed by a Poisson process. Each event has to be met by a decision to either suppress or allow it. The optimization criterion is allowed to depend on the *rate* of decision outcomes within a time interval; the criterion is not necessarily a sum of a loss function over individual decisions.

The problems that we study cannot adequately be modeled as Mavkov or semi-Markov decision problems because the probability of transitioning from any value of decision rates to any other value depends on the exact points in time at which each event occurred in the past. Encoding the entire backlog of time stamps in the state of a Markov process would lead to an unwieldy formalism. The learning formalism which we explore in this paper models the problem directly as a Poisson process with a throttling policy that depends on an explicit data-dependent rate limit, which allows us to refer to a result from queuing theory and derive a convex optimization problem that can be solved efficiently.

Consider the following two scenarios as motivating applications. In order to stage a successful denial-of-service attack, an assailant has to post requests at a rate that exceeds the capacity of the service. A prevention system has to meet each request by a decision to suppress it, or allow it to be processed by the service provider. Suppressing legitimate requests runs up costs. Passing few abusive requests to be processed runs up virtually no costs. Only when the rate of passed abusive requests exceeds a certain capacity, the service becomes unavailable and costs incur. The following second application scenario will serve as a running example throughout this paper. Any email service provider has to deal with a certain fraction of accounts that are set up to disseminate phishing messages and email spam. Serving the occasional spam message causes no harm other than consuming computational ressources. But if the rate of spam messages that an outbound email server discharges triggers alerting mechanisms of other providers, then that outbound server will become blacklisted and the service is disrupted. Naturally, suppressing any legitimate message is a disruption to the service, too.

Let $\mathbf{x}$ denote a sequence of decision events $x_1, \ldots, x_n$; each event is a point $x_i \in \mathcal{X}$ in an instance space. Sequence $\mathbf{t}$ denotes the time stamps $t_i \in \mathbb{R}_+$ of the decision events with $t_i < t_{i+1}$. We define an episode $e$ by the tuple $e = (\mathbf{x}, \mathbf{t}, y)$ which includes a label $y \in \{-1, +1\}$. In our application, an episode corresponds to the sequence of emails sent within an observation interval from a legitimate ($y = -1$) or abusive ($y = +1$) account $e$. We write $\mathbf{x}_i$ and $\mathbf{t}_i$ to denote the initial sequence of the first $i$ elements of $\mathbf{x}$ and $\mathbf{t}$, respectively. Note that the length $n$ of the sequences can be different for different episodes.

Let $\mathcal{A} = \{-1, +1\}$ be a binary decision set, where $+1$ corresponds to suppressing an event and $-1$ corresponds to passing it. The decision model $\pi$ gets to make a decision $\pi(\mathbf{x}_i, \mathbf{t}_i) \in \mathcal{A}$ at each point in time $t_i$ at which an event occurs.

The *outbound rate* $r^\pi(t'|\mathbf{x}, \mathbf{t})$ at time $t'$ for episode $e$ and decision model $\pi$ is a crucial concept. It counts the number of events that were let pass during a time interval of length $\tau$ ending *before* $t'$. It is therefore defined as $r^\pi(t'|\mathbf{x}, \mathbf{t}) = |\{i : \pi(\mathbf{x}_i, \mathbf{t}_i) = -1 \wedge t_i \in [t' - \tau, t')\}|$. In outbound spam throttling, $\tau$ corresponds to the time interval that is used by other providers to estimate the incoming spam rate.

We define an immediate loss function $\ell : Y \times \mathcal{A} \to \mathbb{R}_+$ that specifies the immediate loss of deciding $a \in \mathcal{A}$ for an event with label $y \in Y$ as

$$\ell(y, a) = \begin{cases} c_+ & y = +1 \wedge a = -1 \\ c_- & y = -1 \wedge a = +1 \\ 0 & \text{otherwise,} \end{cases} \tag{1}$$

where $c_+$ and $c_-$ are positive constants, corresponding to costs of false positive and false negative decisions. Additionally, the rate-based loss $\lambda : Y \times \mathbb{R}_+ \to \mathbb{R}_+$ is the loss that runs up per unit of time. We require $\lambda$ to be a convex, monotonically increasing function in the outbound rate for $y = +1$ and to be 0 otherwise. The rate-based loss reflects the risk of the service getting blacklisted based on the current sending behaviour. This risk grows in the rate of spam messages discharged and the duration over which a high sending rate of spam messages is maintained.

The total loss of a model $\pi$ for an episode $e = (\mathbf{x}, \mathbf{t}, y)$ is therefore defined as

$$L(\pi; \mathbf{x}, \mathbf{t}, y) = \int_{t_1}^{t_n + \tau} \lambda(y, r^\pi(t'|\mathbf{x}, \mathbf{t})) \, dt' + \sum_{i=1}^{n} \ell(y, \pi(\mathbf{x}_i, \mathbf{t}_i)) \tag{2}$$

The first term penalizes a high rate of unsuppressed events with label $+1$—in our example, a high rate of unsuppressed spam messages—whereas the second term penalizes each decision individually. For the special case of $\lambda = 0$, the optimization criterion resolves to a risk, and the problem becomes a standard binary classification problem.

An unknown target distribution over $p(\mathbf{x}, \mathbf{t}, y)$ induces the overall optimization goal $\mathbb{E}_{\mathbf{x}, \mathbf{t}, y}[L(\pi; \mathbf{x}, \mathbf{t}, y)]$. The learning problem consists in finding $\pi^* = \operatorname{argmin}_\pi \mathbb{E}_{\mathbf{x}, \mathbf{t}, y}[L(\pi; \mathbf{x}, \mathbf{t}, y)]$ from a training sample of tuples $D = \{(\mathbf{x}_{n^1}^1, \mathbf{t}_{n^1}^1, y^1), \ldots, (\mathbf{x}_{n^m}^m, \mathbf{t}_{n^m}^m, y^m)\}$.

## 2 Poisson Process Model

We assume the following data generation process for episodes $e = (\mathbf{x}, \mathbf{t}, y)$ that will allow us to derive an optimization problem to be solved by the learning procedure. First, a rate parameter $\rho$, label $y$, and the sequence of instances $\mathbf{x}$, are drawn from a joint distribution $p(\mathbf{x}, \rho, y)$. Rate $\rho$ is the parameter of a Poisson process $p(\mathbf{t}|\rho)$ which now generates time sequence $\mathbf{t}$. The expected loss of decision model $\pi$ is taken over all input sequences $\mathbf{x}$, rate parameter $\rho$, label $y$, and over all possible sequences of time stamps $\mathbf{t}$ that can be generated according to the Poisson process.

$$\mathbb{E}_{\mathbf{x}, \mathbf{t}, y}[L(\pi; \mathbf{x}, \mathbf{t}, y)] = \int_{\mathbf{x}} \int_{\mathbf{t}} \int_\rho \sum_y L(\pi; \mathbf{x}, \mathbf{t}, y) p(\mathbf{t}|\rho) p(\mathbf{x}, \rho, y) \, d\rho d\mathbf{t} d\mathbf{x} \tag{3}$$

### 2.1 Derivation of Empirical Loss

In deriving the empirical counterpart of the expected loss, we want to exploit our assumption that time stamps are generated by a Poisson process with unknown but fixed rate parameter. For each

input episode $(\mathbf{x}, \mathbf{t}, y)$, instead of minimizing the expected loss over the single observed sequence of time stamps, we would therefore like to minimize the expected loss over all sequences of time stamps generated by a Poisson process with the rate parameter that has most likely generated the observed sequence of time stamps. Equation 4 introduces the observed time sequence of time stamps $\mathbf{t}'$ into Equation 3 and uses the fact that the rate parameter $\rho$ is independent of $\mathbf{x}$ and $y$ given $\mathbf{t}'$. Equation 5 rearranges the terms, and Equation 6 writes the central integral as a conditional expected value of the loss given the rate $\rho$. Finally, Equation 7 approximates the integral over all values of $\rho$ by a single summand with value $\rho^*$ for each episode.

$$\mathbb{E}_{\mathbf{x},\mathbf{t},y}[L(\pi; \mathbf{x}, \mathbf{t}, y)] = \int_{\mathbf{t}'} \int_{\mathbf{x}} \int_{\mathbf{t}} \int_{\rho} \sum_{y} L(\pi; \mathbf{x}, \mathbf{t}, y) p(\mathbf{t}|\rho) p(\rho|\mathbf{t}') p(\mathbf{x}, \mathbf{t}', y) d\rho d\mathbf{t} d\mathbf{x} d\mathbf{t}' \tag{4}$$

$$= \int_{\mathbf{t}'} \int_{\mathbf{x}} \sum_{y} \left( \int_{\rho} \left( \int_{\mathbf{t}} L(\pi; \mathbf{x}, \mathbf{t}, y) p(\mathbf{t}|\rho) d\mathbf{t} \right) p(\rho|\mathbf{t}') d\rho \right) p(\mathbf{x}, \mathbf{t}', y) d\mathbf{x} d\mathbf{t}' \tag{5}$$

$$= \int_{\mathbf{t}'} \int_{\mathbf{x}} \sum_{y} \left( \int_{\rho} \left( \mathbb{E}_{\mathbf{t}} \left[ L(\pi; \mathbf{x}, \mathbf{t}, y) \mid \rho \right] p(\rho|\mathbf{t}') d\rho \right) \right) p(\mathbf{x}, \mathbf{t}', y) d\mathbf{x} d\mathbf{t}' \tag{6}$$

$$\approx \int_{\mathbf{t}'} \int_{\mathbf{x}} \sum_{y} \mathbb{E}_{\mathbf{t}} \left[ L(\pi; \mathbf{x}, \mathbf{t}, y) \mid \rho^* \right] p(\mathbf{x}, \mathbf{t}', y) d\mathbf{x} d\mathbf{t}' \tag{7}$$

We arrive at the regularized risk functional in Equation 8 by replacing $p(\mathbf{x}, \mathbf{t}', y)$ by $\frac{1}{m}$ for all observations in $D$ and inserting MAP estimate $\rho_e^*$ as parameter that generated time stamps $\mathbf{t}^e$. The influence of the convex regularizer $\Omega$ is determined by regularization parameter $\eta > 0$.

$$\hat{\mathbb{E}}_{\mathbf{x},\mathbf{t},y}[L(\pi; \mathbf{x}, \mathbf{t}, y)] = \frac{1}{m} \sum_{e=1}^{m} \mathbb{E}_{\mathbf{t}} \left[ L(\pi; \mathbf{x}^e, \mathbf{t}, y^e) \mid \rho_e^* \right] + \eta \Omega(\pi) \tag{8}$$

$$\text{with} \qquad \rho_e^* = \text{argmax}_{\rho} p(\rho|\mathbf{t}^e)$$

Minimizing this risk functional is the basis of the learning procedure in the next section. As noted in Section 1, for the special case when the rate-based loss $\lambda$ is zero, the problem reduces to a standard weighted binary classification problem and would be easy to solve with standard learning algorithms. However, as we will see in Section 4, the $\lambda$-dependent loss makes the task of learning a decision function hard to solve; attributing individual decisions with their "fair share" of the rate loss—and thus estimating the cost of the decision—is problematic. The Erlang learning model of Section 3 employs a decision function that allows to factorize the rate loss naturally.

## 3    Erlang Learning Model

In the following we derive an optimization problem that is based on modeling the policy as a data-dependent rate limit. This allows us to apply a result from queuing theory and approximate the empirical risk functional of Equation (8) with a convex upper bound. We define decision model $\pi$ in terms of the function $f_\theta(\mathbf{x}_i, \mathbf{t}_i) = \exp(\theta^\mathsf{T} \phi(\mathbf{x}_i, \mathbf{t}_i))$ which sets a limit on the admissible rate of events, where $\phi$ is some feature mapping of the initial sequence $(\mathbf{x}_i, \mathbf{t}_i)$ and $\theta$ is a parameter vector. The throttling model is defined as

$$\pi(\mathbf{x}_i, \mathbf{t}_i) = \begin{cases} -1 \ (\text{"allow"}) & \text{if } r^\pi(t_i|\mathbf{x}_i, \mathbf{t}_i) + 1 \leq f_\theta(\mathbf{x}_i, \mathbf{t}_i) \\ +1 \ (\text{"suppress"}) & \text{otherwise.} \end{cases} \tag{9}$$

The decision model blocks event $x_i$, if the number of instances that were sent within $[t_i - \tau, t_i)$, plus the current instance, would exceed rate limit $f_\theta(\mathbf{x}_i, \mathbf{t}_i)$. We will now transform the optimization goal of Equation 8 into an optimization problem that can be solved by standard convex optimization tools. To this end, we first decompose the expected loss of an input sequence given the rate parameter in Equation 8 into immediate and rate-dependent loss terms. Note that $\mathbf{t}^e$ denotes the observed training sequence whereas $\mathbf{t}$ serves as expectation variable for the expectation $\mathbb{E}_{\mathbf{t}}[\cdot|\rho_e^*]$ over all sequences

conditional on the Poisson process rate parameter $\rho_e^*$ as in Equation 8.

$$\mathbb{E}_{\mathbf{t}}\left[L(\pi; \mathbf{x}^e, \mathbf{t}, y^e) \mid \rho_e^*\right]$$

$$= \mathbb{E}_{\mathbf{t}}\left[\int_{t_1}^{t_{n^e}+\tau} \lambda\left(y^e, r^\pi(t'|\mathbf{x}^e, \mathbf{t})\right) dt' \mid \rho_e^*\right] + \sum_{i=1}^{n^e} \mathbb{E}_{\mathbf{t}}[\ell(y^e, \pi(\mathbf{x}_i^e, \mathbf{t}_i)) \mid \rho_e^*] \qquad (10)$$

$$= \mathbb{E}_{\mathbf{t}}\left[\int_{t_1}^{t_{n^e}+\tau} \lambda\left(y^e, r^\pi(t'|\mathbf{x}^e, \mathbf{t})\right) dt' \mid \rho_e^*\right] + \sum_{i=1}^{n^e} \mathbb{E}_{\mathbf{t}}\left[\delta\left(\pi(\mathbf{x}_i^e, \mathbf{t}_i) \neq y^e\right) \mid \rho_e^*\right] \ell(y^e, -y^e) \quad (11)$$

Equation 10 uses the definition of the loss function in Equation 2. Equation 11 exploits that only decisions against the correct label, $\pi(\mathbf{x}_i^e, \mathbf{t}_i) \neq y^e$, incur a positive loss $\ell(y, \pi(\mathbf{x}_i^e, \mathbf{t}_i))$.

We will first derive a convex approximation of the expected rate-based loss $\mathbb{E}_{\mathbf{t}}[\int_{t_1}^{t_{n^e}+\tau} \lambda\left(y^e, r^\pi(t'|\mathbf{x}^e, \mathbf{t})\right) dt'|\rho_e^*]$ (left side of Equation 11). Our definition of the decision model allows us to factorize the expected rate-based loss into contributions of individual rate limit decisions. The convexity will be addressed by Theorem 1.

Since the outbound rate $r^\pi$ increases only at decision points $t_i$, we can upper-bound its value with the value immediately after the most recent decision in Equation 12. Equation 13 approximates the actual outbound rate with the rate limit given by $f_\theta(\mathbf{x}_i^e, \mathbf{t}_i^e)$. This is reasonable because the outbound rate depends on the policy decisions which are defined in terms of the rate limit. Because $\mathbf{t}$ is generated by a Poisson process, $\mathbb{E}_{\mathbf{t}}[t_{i+1} - t_i \mid \rho_e^*] = \frac{1}{\rho_e^*}$ (Equation 14).

$$\mathbb{E}_{\mathbf{t}}\left[\int_{t_1}^{t_{n^e}+\tau} \lambda\left(y^e, r^\pi(t'|\mathbf{x}^e, \mathbf{t})\right) dt' \mid \rho_e^*\right]$$

$$\leq \sum_{i=1}^{n^e-1} \mathbb{E}_{\mathbf{t}}[t_{i+1} - t_i \mid \rho_e^*] \lambda(y^e, r^\pi(t_i|\mathbf{x}^e, \mathbf{t})) + \tau\lambda(y^e, r^\pi(t_{n^e}|\mathbf{x}^e, \mathbf{t})) \qquad (12)$$

$$\approx \sum_{i=1}^{n^e-1} \mathbb{E}_{\mathbf{t}}[t_{i+1} - t_i \mid \rho_e^*] \lambda\left(y^e, f_\theta(\mathbf{x}_i^e, \mathbf{t}_i^e)\right) + \tau\lambda\left(y^e, f_\theta(\mathbf{x}_{n^e}^e, \mathbf{t}_{n^e}^e)\right) \qquad (13)$$

$$= \sum_{i=1}^{n^e-1} \frac{1}{\rho_e^*} \lambda\left(y^e, f_\theta(\mathbf{x}_i^e, \mathbf{t}_i^e)\right) + \tau\lambda\left(y^e, f_\theta(\mathbf{x}_{n^e}^e, \mathbf{t}_{n^e}^e)\right) \qquad (14)$$

We have thus established a convex approximation of the left side of Equation 11.

We will now derive a closed form approximation of $\mathbb{E}_{\mathbf{t}}[\delta\left(\pi(\mathbf{x}_i^e, \mathbf{t}_i) \neq y^e\right) \mid \rho_e^*]$, the second part of the loss functional in Equation 11. Queuing theory provides a convex approximation: The *Erlang-B* formula [5] gives the probability that a queuing process which maintains a constant rate limit of $f$ within a time interval of $\tau$ will block an event when events are generated by a Poisson process with given rate parameter $\rho$. Fortet's formula (Equation 15) generalizes the Erlang-B formula for non-integer rate limits.

$$B(f, \rho\tau) = \frac{1}{\int_0^\infty e^{-z}(1 + \frac{z}{\rho\tau})^f dz} \qquad (15)$$

The integral can be computed efficiently using a rapidly converging series, c.f. [5]. The formula requires a constant rate limit, so that the process can reach an equilibrium. In our model, the rate limit $f_\theta(\mathbf{x}_i, \mathbf{t}_i)$ is a function of the sequences $\mathbf{x}_i$ and $\mathbf{t}_i$ until instance $x_i$, and Fortet's formula therefore serves as an approximation.

$$\mathbb{E}_{\mathbf{t}}\left[\delta(\pi(\mathbf{x}_i^e, \mathbf{t}_i) = 1)|\rho_e^*\right] \approx B(f_\theta(\mathbf{x}_i^e, \mathbf{t}_i^e), \rho_e^*\tau) \qquad (16)$$

$$= \left[\int_0^\infty e^{-z}(1 + \frac{z}{\rho_e^*\tau})^{f_\theta(\mathbf{x}_i^e, \mathbf{t}_i^e)} dz\right]^{-1} \qquad (17)$$

Unfortunately, Equation 17 is not convex in $\theta$. We approximate it with the convex upper bound $-\log\left(1 - B(f_\theta(\mathbf{x}_i^e, \mathbf{t}_i^e), \rho_e^*\tau)\right)$ (cf. the dashed green line in the left panel of Figure 2(b) for an illustration). This is an upper bound, because $-\log p \geq 1 - p$ for $0 \leq p \leq 1$; its convexity is addressed by Theorem 1. Likewise, $\mathbb{E}_{\mathbf{t}}\left[\delta(\pi(\mathbf{x}_i^e, \mathbf{t}_i) = -1)|\rho_e^*\right]$ is approximated by upper bound $\log\left(B(f_\theta(\mathbf{x}_i^e, \mathbf{t}_i^e), \rho_e^*\tau)\right)$. We have thus derived a convex upper bound of $\mathbb{E}_{\mathbf{t}}[\delta\left(\pi(\mathbf{x}_i^e, \mathbf{t}_i) \neq y^e\right)|\rho_e^*]$.

Combining the two components of the optimization goal (Equation 11) and adding convex regularizer $\Omega(\theta)$ and regularization parameter $\eta > 0$ (Equation 8), we arrive at an optimization problem for finding the optimal policy parameters $\theta$.

**Optimization Problem 1** (Erlang Learning Model). *Over $\theta$, minimize*

$$
\begin{aligned}
R(\theta) = \frac{1}{m} \sum_{e=1}^{m} \bigg\{ & \sum_{i=1}^{n^e-1} \frac{1}{\rho_e{}^*} \lambda\big(y^e, f_\theta(\mathbf{x}_i^e, \mathbf{t}_i^e)\big) + \tau \lambda\big(y^e, f_\theta(\mathbf{x}_{n^e}^e, \mathbf{t}_{n^e}^e)\big) \\
& + \sum_{i=1}^{n^e} - \log\big[\delta(y^e{=}1) - y^e B\big(f_\theta(\mathbf{x}_i^e, \mathbf{t}_i^e), \rho_e^* \tau\big)\big] \ell(y^e, -y^e) \bigg\} + \eta\Omega(\theta)
\end{aligned} \tag{18}
$$

Next we show that minimizing risk functional $R$ amounts to solving a convex optimization problem.

**Theorem 1** (Convexity of $R$). *$R(\theta)$ is a convex risk functional in $\theta$ for any $\rho_e^* > 0$ and $\tau > 0$.*

*Proof.* The convexity of $\lambda$ and $\Omega$ follows from their definitions. It remains to be shown that both $-\log B(f_\theta(\cdot), \rho_e^*\tau))$ and $-\log(1 - B(f_\theta(\cdot), \rho_e^*)$ are convex in $\theta$. Component $\ell(y^e, -y^e)$ of Equation 18 is independent of $\theta$. It is known that Fortet's formula $B(f, \rho_e^*\tau))$ is convex, monotically decreasing, and positive in $f$ for $\rho_e^*\tau > 0$ [5]. Furthermore $-\log(B(f, \rho_e^*\tau)))$ is convex and monotonically increasing. Since $f_\theta(\cdot)$ is convex in $\theta$, it follows that $-\log(B(f_\theta(\cdot), \rho_e^*))$ is also convex. Next, we show that $-\log(1 - B(f_\theta(\cdot), \rho_e^*\tau)))$ is convex and monotonically decreasing. From the above it follows that $b(f) = 1 - B(f, \rho_e^*\tau))$ is monotonically increasing, concave and positive. Therefore, $\frac{d^2}{df^2} - \ln(b(f)) = \frac{1}{b^2(f)} b'(f) + b''(f) \frac{-1}{b(f)} \geq 0$ as both summands are positive. Again, it follows that $-\log(1 - B(f_\theta(\cdot), \rho_e^*\tau)))$ is convex in $\theta$ due to the definition of $f_\theta$. $\qquad\square$

## 4 Prior Work and Reference Methods

We will now discuss how the problem of minimizing the expected loss, $\pi^* = \operatorname{argmin}_\pi \mathbb{E}_{\mathbf{x},\mathbf{t},y}[L(\pi; \mathbf{x}, \mathbf{t}, y)]$, from a sample of sequences $\mathbf{x}$ of events with labels $y$ and observed rate parameters $\rho^*$ relates to previously studied methods. Sequential decision-making problems are commonly solved by reinforcement learning approaches, which have to attribute the loss of an episode (Equation 2) to individual decisions in order to learn to decide optimally in each state. Thus, a crucial part of defining an appropriate procedure for learning the optimal policy consists in defining an appropriate state-action loss function. $Q^\pi(s, a)$ estimates the loss of performing action $a$ in state $s$ when following policy $\pi$ for the rest of the episode.

Several different state-action loss functions for related problems have been investigated in the literature. For example, policy gradient methods such as in [4] assign the loss of an episode to individual decisions proportional to the log-probabilities of the decisions. Other approaches use sampled estimates of the rest of the episode $Q(s_i, a_i) = L(\pi, \mathbf{s}) - L(\pi, \mathbf{s}_i)$ or the expected loss if a distribution of states of the episode is known [7]. Such general purpose methods, however, are not the optimal choice for the particular problem instance at hand. Consider the special case $\lambda = 0$, where the problem reduces to a sequence of independent binary decisions. Assigning the cumulative loss of the episode to all instances leads to a grave distortion of the optimization criterion.

As reference in our experiments we use a state-action loss function that assigns the immediate loss $\ell(y, a_i)$ to state $s_i$ only. Decision $a_i$ determines the loss incurred by $\lambda$ only for $\tau$ time units, in the interval $[t_i, t_i + \tau)$. The corresponding rate loss is $\int_{t_i}^{t_i+\tau} \lambda(y, r^\pi(t'|\mathbf{x}, \mathbf{t})) dt'$. Thus, the loss of deciding $a_i = -1$ instead of $a_i = +1$ is the difference in the corresponding $\lambda$-induced loss. Let $\mathbf{x}^{-i}, \mathbf{t}^{-i}$ denote the sequence $\mathbf{x}, \mathbf{t}$ without instance $x_i$. This leads to a state-action loss function that is the sum of immediate loss and $\lambda$-induced loss; it serves as our first baseline.

$$
Q_{it}^\pi(s_i, a) = \ell(y, a) + \delta(a{=}{-}1) \int_{t_i}^{t_i+\tau} \lambda(y, r^\pi(t'|\mathbf{x}^{-i}, \mathbf{t}^{-i}) + 1) - \lambda(y, r^\pi(t'|\mathbf{x}^{-i}, \mathbf{t}^{-i})) dt' \tag{19}
$$

By approximating $\int_{t_i}^{t_i+\tau} \lambda(y, r^\pi(t'|\mathbf{x}, \mathbf{t}))$ with $\tau \lambda(y, r^\pi(t_i|\mathbf{x}, \mathbf{t}))$, we define the state-action loss function of a second plausible state-action loss that, instead of using the observed loss to estimate

the loss of an action, approximates it with the loss that would be incurred by the current outbound rate $r^\pi(t_i|\mathbf{x}^{-i}, \mathbf{t}^{-i})$ for $\tau$ time units.

$$Q_{ub}^\pi(s_i, a) = \ell(y, a) + \delta(a = -1) \left[ \tau \big( \lambda(y, r^\pi(t_i|\mathbf{x}^{-i}, \mathbf{t}^{-i}) + 1) - \lambda(y, r^\pi(t_i|\mathbf{x}^{-i}, \mathbf{t}^{-i})) \big) \right] \quad (20)$$

The state variable $s$ has to encode all information a policy needs to decide. Since the loss crucially depends on outbound rate $r^\pi(t'|\mathbf{x}, \mathbf{t})$, any throttling model must have access to the current outbound rate. The transition between a current and a subsequent rate depends on the time at which the next event occurs, but also on the entire backlog of events, because past events may drop out of the interval $\tau$ at any time. In analogy to the information that is available to the Erlang learning model, it is natural to encode states $s_i$ as a vector of features $\phi(\mathbf{x}_i, \mathbf{t}_i)$ (see Section 5 for details) together with the current outbound rate $r^\pi(t_i|\mathbf{x}, \mathbf{t})$. Given a representation of the state and a state-action loss function, different approaches for defining the policy $\pi$ and optimizing its parameters have been investigated. For our baselines, we use the following two methods.

**Policy gradient.**   Policy gradient methods model a stochastic policy directly as a parameterized decision function. They perform a gradient descent that always converges to a local optimum [8]. The gradient of the expected loss with respect to the parameters is estimated in each iteration $k$ for the distribution over episodes, states, and losses that the *current* policy $\pi_k$ induces. However, in order to achieve fast convergence to the optimal polity, one would need to determine the gradient for the distribution over episodes, states, and losses induced by the *optimal* policy. We implement two policy gradient algorithms for experimentation which only differ in using $Q_{it}$ and $Q_{ub}$, respectively. They are denoted $\mathrm{PG}_{it}$ and $\mathrm{PG}_{ub}$ in the experiments. Both use a logistic regression function as decision function, the two-class equivalent of the Gibbs distribution which is used in the literature.

**Iterative Classifier.**   The second approach is to represent policies as classifiers and to employ methods for supervised classification learning. A variety of papers addresses this approach [6, 3, 7]. We use an algorithm that is inspired by [1, 2] and is adapted to the problem setting at hand. Blatt and Hero [2] investigate an algorithm that finds non-stationary policies for two-action T-step MDPs by solving a sequence of one-step decisions via a binary classifier. Classifiers $\pi_t$ for time step $t$ are learned iteratively on the distribution of states generated by the policy $(\pi_0, \ldots, \pi_{t-1})$. Our derived algorithm iteratively learns weighted support vector machine (SVM) classifier $\pi_{k+1}$ in iteration $k+1$ on the set of instances and losses $Q^{\pi_k}(s, a)$ that were observed after classifier $\pi_k$ was used as policy on the training sample. The weight vector of $\pi_k$ is denoted $\theta_k$. The weight of misclassification of $s$ is given by $Q^{\pi_k}(s, -y)$. The SVM weight vector is altered in each iteration as $\theta_{k+1} = (1 - \alpha_k)\theta_k + \alpha_k\hat{\theta}$, where $\hat{\theta}$ is the weight vector of the new classifier that was learned on the observed losses. In the experiments, two iterative SVM learner were implemented, denoted It-$\mathrm{SVM}_{it}$ and It-$\mathrm{SVM}_{ub}$, corresponding to the used state-action losses $Q_{it}$ and $Q_{ub}$, respectively. Note that for the special case $\lambda = 0$ the iterative SVM algorithm reduces to a standard SVM algorithm.

All four procedures iteratively estimate the loss of a policy decision on the data via a state-action loss function and learn a new policy $\pi$ based on this estimated cost of the decisions. Convergence guarantees typically require the Markov assumption; that is, the process is required to possess a stationary transition distribution $P(s_{i+1}|s_i, a_i)$. Since the transition distribution in fact depends on the entire backlog of time stamps and the duration over which state $s_i$ has been maintained, the Markov assumption is violated to some extent in practice. In addition to that, $\lambda$-based loss estimates are sampled from a Poisson process. In each iteration $\pi$ is learned to minimize sampled and inherently random losses of decisions. Thus, convergence to a robust solution becomes unlikely. In contrast, the Erlang learning model directly minimizes the $\lambda$-loss by assigning a rate limit. The rate limit implies an expectation of decisions. In other words, the $\lambda$-based loss is minimized without explicitly estimating the loss of any decisions that are implied by the rate limit. The convexity of the risk functional in Optimization Problem 1 guarantees convergence to the global optimum.

# 5   Application

The goal of our experiments is to study the relative benefits of the Erlang learning model and the four reference methods over a number of loss functions. The subject of our experimentation is the problem of suppressing spam and phishing messages sent from abusive accounts registered at a large email service provider. We sample approximately 1,000,000 emails sent from approximately

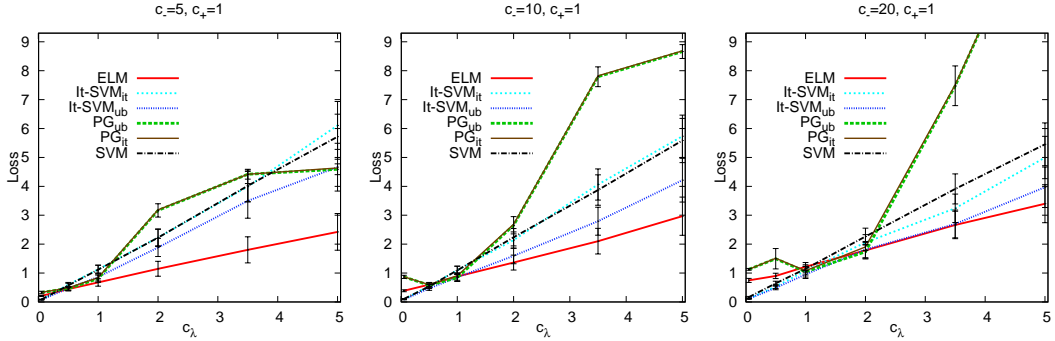

Figure 1: Average loss on test data depending on the influence of the rate loss $c_\lambda$ for different immediate loss constants $c_-$ and $c_+$.

10,000 randomly selected accounts over two days and label them automatically based on information passed by other email service providers via feedback loops (in most cases triggered by "report spam" buttons). Because of this automatic labeling process, the labels contain a certain smount of noise.

Feature mapping $\phi$ determines a vector of moving average and moving variance estimates of several attributes of the email stream. These attributes measure the frequency of subject changes and sender address changes, and the number of recipients. Other attributes indicate whether the subject line or the sender address have been observed before within a window of time. Additionally, a moving average estimate of the rate $\rho$ is used as feature. Finally, other attributes quantify the size of the message and the score returned by a content-based spam filter employed by the email service.

We implemented the baseline methods that were descibed in Section 4, namely the iterative SVM methods It-SVM$_{ub}$ and It-SVM$_{it}$ and the policy gradient methods PG$_{ub}$ and PG$_{it}$. Additionally, we used a standard support vector machine classifier SVM with weights of misclassification corresponding to the costs defined in Equation 1. The Erlang learning model is denoted ELM in the plots. Linear decision functions were used for all baselines.

In our experiments, we assume a cost that is quadratic in the outbound rate. That is, $\lambda(1, r^\pi(t'|\mathbf{x}, \mathbf{t}))) = c_\lambda \cdot r^\pi(t'|\mathbf{x}, \mathbf{t})^2$ with $c_\lambda > 0$ determining the influence of the rate loss to the overall loss. The time interval $\tau$ was chosen to be 100 seconds. Regularizer $\Omega(\theta)$ as in Optimization problem 1 is the commonly used squared $l_2$-norm $\Omega(\theta) = \|\theta\|_2^2$.

We evaluated our method for different costs of incorrectly classified non-spam emails ($c_-$), incorrectly classified spam emails ($c_+$) (see the definition of $\ell$ in Equation 1), and rate of outbound spam messages ($c_\lambda$). For each setting, we repeated 100 runs; each run used about 50%, chosen at random, as training data and the remaining part as test data. Splits where chosen such that there were equally many spam episodes in training and test set. We tuned the regularization parameter $\eta$ for the Erlang learning model as well as the corresponding regularization parameters of the iterative SVM methods and the standard SVM on a separate tuning set that was split randomly from the training data.

## 5.1 Results

Figure 1 shows the resulting average loss of the Erlang learning model and reference methods. Each of the three plots shows loss versus parameter $c_\lambda$ which determines the influence of the rate loss on the overall loss. The left plot shows the loss for $c_- = 5$ and $c_+ = 1$, the center plot for ($c_- = 10, c_+ = 1$), and the right plot for ($c_- = 20, c_+ = 1$).

We can see in Figure 1 that the Erlang learning model outperforms all baseline methods for larger values of $c_\lambda$—more influence of the rate dependent loss on the overall loss—in two of the three settings. For $c_- = 20$ and $c_+ = 1$ (right panel), the performance is comparable to the best baseline method It-SVM$_{ub}$; only for the largest shown $c_\lambda = 5$ does the ELM outperform this baseline. The iterative classifier It-SVM$_{ub}$ that uses the approximated state-action loss $Q_{ub}$ performs uniformly better than It-SVM$_{it}$, the iterative SVM method that uses the sampled loss from the previous iteration. It-SVM$_{it}$ itself surprisingly shows very similar performance to that of the standard SVM method; only for the setting $c_- = 20$ and $c_+ = 1$ in the right panel does this iterative SVM method show superior performance. Both policy gradient methods perform comparable to the Erlang learning model for smaller values of $c_\lambda$ but deteriorate for larger values.

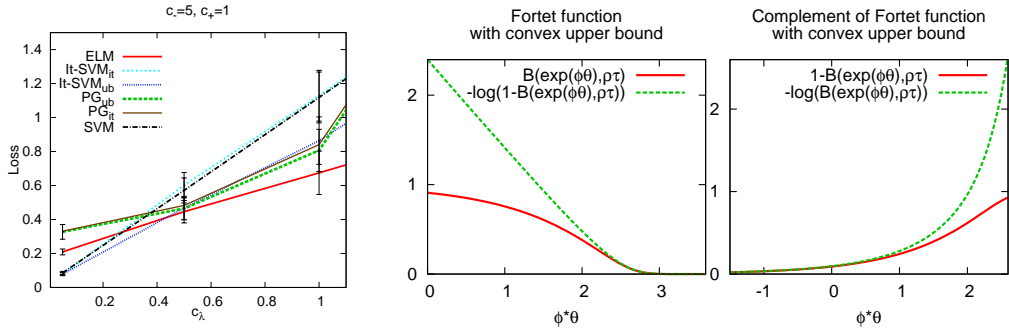

(a) Average loss and standard error for small values of $c_\lambda$.

(b) Left: Fortet's formula $B(e^{\phi\theta}, \rho\tau)$ (Equation 17) and its upper bound $-\log(1 - B(e^{\phi\theta}, \rho))$ for $\rho\tau = 10$. Right: $1 - B(e^{\phi\theta}, \rho)$ and respective upper bound $-\log(B(e^{\phi\theta}, \rho))$.

As expected, the iterative SVM and the standard SVM algorithms perform better than the Erlang learning model and policy gradient models if the influence of the rate pedendent loss is very small. This can best be seen in Figure 2(a). It shows a detail of the results for the setting $c_- = 5$ and $c_+ = 1$, for $c_\lambda$ ranging only from 0 to 1. This is the expected outcome following the considerations in Section 4. If $c_\lambda$ is close to 0, the problem approximately reduces to a standard binary classification problem, thus favoring the very good classification performance of support vector machines. However, for larger $c_\lambda$ the influence of the rate dependent loss rises and more and more dominates the immediate classification loss $\ell$. Consequently, for those cases — which are the important ones in this real world application — the better rate loss estimation of the Erlang learning model compared to the baselines leads to better performance.

The average training times for the Erlang learning model and the reference methods are in the same order of magnitude. The SVM algorithm took 14 minutes in average to converge to a solution. The Erlang learning model converged after 44 minutes and the policy gradient methods took approximately 45 minutes. The training times of the iterative classifier methods were about 60 minutes.

## 6  Conclusion

We devised a model for sequential decision-making problems in which events are generated by a Poisson process and the loss may depend on the rate of decision outcomes. Using a throttling policy that enforces a data-dependent rate-limit, we were able to factor the loss over single events. Applying a result from queuing theory led us to a closed-form approximation of the immediate event-specific loss under a rate limit set by a policy. Both parts led to a closed-form convex optimization problem. Our experiments explored the learning model for the problem of suppressing abuse of an email service. We observed significant improvements over iterative reinforcement learning baselines. The model is being employed to this end in the email service provided by web hosting firm STRATO. It has replaced a procedure of manual deactivation of accounts after inspection triggered by spam reports.

### Acknowledgments

We gratefully acknowledge support from STRATO Rechenzentrum AG and the German Science Foundation DFG.

## References

[1] J.A. Bagnell, S. Kakade, A. Ng, and J. Schneider. Policy search by dynamic programming. *Advances in Neural Information Processing Systems*, 16, 2004.

[2] D. Blatt and A.O. Hero. From weighted classification to policy search. *Advances in Neural Information Processing Systems*, 18, 2006.

[3] C. Dimitrakakis and M.G. Lagoudakis. Rollout sampling approximate policy iteration. *Machine Learning*, 72(3):157–171, 2008.

[4] M. Ghavamzadeh and Y. Engel. Bayesian policy gradient algorithms. *Advances in Neural Information Processing Systems*, 19, 2007.

[5] D.L. Jagerman, B. Melamed, and W. Willinger. Stochastic modeling of traffic processes. *Frontiers in queueing: models, methods and problems*, pages 271–370, 1996.

[6] M.G. Lagoudakis and R. Parr. Reinforcement learning as classification: Leveraging modern classifiers. In *Proceedings of the 20th International Conference on Machine Learning*, 2003.

[7] J. Langford and B. Zadrozny. Relating reinforcement learning performance to classification performance. In *Proceedings of the 22nd International Conference on Machine learning*, 2005.

[8] R.S. Sutton, D. McAllester, S. Singh, and Y. Mansour. Policy gradient methods for reinforcement learning with function approximation. *Advances in Neural Information Processing Systems*, 12, 2000.

